# Factorial Switching Kalman Filters for Condition Monitoring in Neonatal Intensive Care

**Christopher K. I. Williams and John Quinn**
School of Informatics, University of Edinburgh
Edinburgh EH1 2QL, UK
c.k.i.williams@ed.ac.uk
john.quinn@ed.ac.uk

**Neil McIntosh**
Simpson Centre for Reproductive
Health, Edinburgh EH16 4SB, UK
neil.mcintosh@ed.ac.uk

## Abstract

The observed physiological dynamics of an infant receiving intensive care are affected by many possible factors, including interventions to the baby, the operation of the monitoring equipment and the state of health. The Factorial Switching Kalman Filter can be used to infer the presence of such factors from a sequence of observations, and to estimate the true values where these observations have been corrupted. We apply this model to clinical time series data and show it to be effective in identifying a number of artifactual and physiological patterns.

## 1 Introduction

In a neonatal intensive care unit (NICU), an infant's vital signs, including heart rate, blood pressures, blood gas properties and temperatures, are continuously monitored and displayed at the cotside. The levels of these measurements and the way they vary give an indication of the baby's health, but they can be affected by many different things. The potential factors include handling of the baby, different cardiovascular and respiratory conditions, the effects of drugs which have been administered, and the setup of the monitoring equipment. Each factor has an effect on the dynamics of the observations, some by affecting the physiology of the baby (such as an oxygen desaturation), and some by overwriting the measurements with artifactual values (such as a probe dropout).

We use a Factorial Switching Kalman Filter (FSKF) to model such data. This consists of three sets of variables which we call *factors*, *state* and *observations*, as indicated in Figure 1(a). There are a number of hidden factors; these are discrete variables, modelling for example if the baby is in a normal respiratory state or not, or if a probe is disconnected or not. The state of baby denotes continuous-valued quantities; this models the true values of infant's physiological variables, but also has dimensions to model certain artifact processes (see below). The observations are those readings obtained from the monitoring equipment, and are subject to corruption by artifact etc.

By describing the dynamical regime associated with each combination of factors as a linear Gaussian model we obtain a FSKF, which extends the Switching Kalman Filter (see e.g. [10, 3]) to incorporate multiple independent factors. With this method we can infer the

value of each factor and estimate the true values of vital signs during the times that the measurements are obscured by artifact. By using an interpretable hidden state structure for this application, domain knowledge can be used to set some of the parameters.

This paper demonstrates an application of the FSKF to NICU monitoring data. In Section 2 we introduce the model, and discuss the links to previous work in the field. In Section 3 we describe an approach for setting the parameters of the model and in Section 4 we show results from the model when applied to NICU data. Finally we close with a discussion in Section 5.

## 2  Model description

The Factorial Switching Kalman Filter is shown in Figure 1(a). In this model, $M$ factors $f_t^{(1)} \ldots f_t^{(M)}$ affect the hidden continuous state $\mathbf{x}_t$ and the observations $\mathbf{y}_t$. The factor $f^{(m)}$ can take on $K^{(m)}$ different values. For example, a simple factor is 'ECG probe dropout', taking on two possible values, 'dropped out' or 'normal'. As factors in this application can affect the observations either by altering the baby's physiology or overwriting them with artifactual values, the hidden state vector $\mathbf{x}_t$ contains information on both the 'true' physiological condition of the baby and on the levels of any artifactual processes.

The dynamical regime at time $t$ is controlled by the 'switch' variable $s_t$, which is the cross product of the individual factors,

$$s_t = f_t^{(1)} \otimes \ldots \otimes f_t^{(M)} \ . \tag{1}$$

For a given setting of $s_t$, the hidden continuous state and the observations are related by:

$$\mathbf{x}_t \sim \mathcal{N}(A(s_t)\mathbf{x}_{t-1} + \mathbf{d}(s_t), Q(s_t)), \qquad \mathbf{y}_t \sim \mathcal{N}(H(s_t)\mathbf{x}_t, R(s_t)), \tag{2}$$

where as in the SKF the system dynamics and observation process are dependent on the switch variable. Here $A(s_t)$ is a square system matrix, $\mathbf{d}(s_t)$ is a drift vector, $H(s_t)$ is the state-observations matrix, and $Q(s_t)$ and $R(s_t)$ are noise covariance matrices. The factors are taken to be a priori independent and first-order Markovian, so that

$$p(s_t|s_{t-1}) = \prod_{m=1}^{M} p(f_t^{(m)}|f_{t-1}^{(m)}) \ . \tag{3}$$

### 2.1  Application-specific setup

The continuous hidden state vector $\mathbf{x}$ contains two types of values, the true physiological values, $\mathbf{x}_p$, and those of artifactual processes, $\mathbf{x}_a$. The true values are modelled as independent autoregressive processes, described in more detail in section 3. To represent this as a state space, the vector $\mathbf{x}_t$ has to contain the value of the current state and store the value of the states at previous times.

Note that artifact state values can be affected by physiological state, but not the other way round. For example, one factor we model is the arterial blood sample, seen in Figure 1(b), lower panel. This occurs when a three-way valve is closed in the baby's arterial line, in order for a clinician to draw blood for a sample. While the valve is closed a pump works against the pressure sensor, causing the systolic and diastolic blood pressure measurements to rise artificially. The artifactual values in this case always start at around the value of the baby's diastolic blood pressure.

The factors modelled in these experiments are listed in Table 1. The dropout factors represent the case where probes are disconnected and measurements fall to zero on the channels supplied by that probe. In this case, the true physiological values are completely hidden.

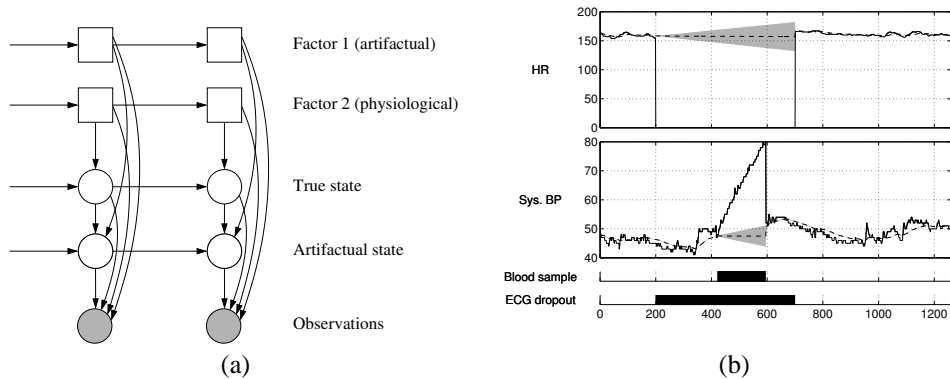

(a)                                         (b)

Figure 1: (a) shows a graphical representation of a Factorial Switching Kalman Filter, with $M = 2$ factors. Squares are discrete values, circles are continuous and shaded nodes are observed. Panel (b) shows ECG dropout and arterial blood sample events occurring simultaneously. HR denotes heart rate, Sys. BP denotes the systolic blood pressure, and times are in seconds. The dashed line indicates the estimate of true values and the greyscale denotes two standard deviation error bars. We see uncertainty increasing while observations are artifactual. The traces at the bottom show the inferred duration of the arterial blood sample and ECG dropout events.

The transcutaneous probe (TCP) provides measurements of the partial pressure of oxygen ($TcPO_2$) and carbon dioxide ($TcPCO_2$) in the baby's blood, and is recalibrated every few hours. This process has three stages: firstly calibration, where $TcPO_2$ and $TcPCO_2$ are set to known values by applying a gas to the probe, secondly a stage where the probe is in air and $TcPCO_2$ drops to zero, and finally an equilibration phase where both values slowly return to the physiological baseline when the probe is replaced.

As explained above, when an arterial blood sample is being taken one sees a characteristic ramp in the blood pressure measurements. Temperature probe disconnection frequently occurs in conjunction with handling. The core temperature probe is under the baby and can come off when the baby is turned over for an examination, causing the readings to drop to the ambient temperature level of the incubator over the course of a few minutes. When the probe is reapplied, the measurements gradually return to the true level of the baby's core temperature.

Bradycardia is a genuine physiological occurrence where the heart rate temporarily drops, often with a characteristic curve, then a systemic reaction brings the measurements back to the baseline. The final factor models opening of the portals on the baby's incubator. Because the environment within the incubator is closely regulated, an intervention can be inferred from a fall in the incubator humidity measurements. While the portals are open and a clinician is handling the baby, we expect increased variability in the measurements from the probes that are still attached.

## 2.2 Inference

For the application of real time clinical monitoring, we are interested in filtering, inferring $\mathbf{x}_t$ and $s_t$ from the observations $\mathbf{y}_{1:t}$. However, the time taken for exact inference of the posterior $p(\mathbf{x}_t, s_t | \mathbf{y}_{1:t})$ scales exponentially with $t$, making it intractable. This is because the probabilities of having moved between every possible combination of switch settings in times $t - 1$ and $t$ are needed to calculate the posterior at time $t$. Hence the number of

| FACTOR | POSSIBLE SETTINGS |
|---|---|
| 5 Probe dropout factors: pulse oximeter, ECG, arterial line, temperature probe, transcutaneous probe | **1**. Dropped out   **2**. Normal |
| TCP recalibration | **1**. $O_2$ high, $CO_2$ low   **2**. $CO_2 \to 0$<br>**3**. Equilibration   **4**. Normal |
| Arterial blood sample | **1**. Blood sample   **2**. Normal |
| Temperature probe disconnection | **1**. Temperature probe disconnection<br>**2**. Reconnection   **3**. Normal |
| Bradycardia | **1**. Bradycardia onset<br>**2**. HR restabilisation   **3**. Normal |
| Incubator open | **1**. Incubator portals opened<br>**2**. Normal |

Table 1: Description of factors.

Gaussians needed to represent the posterior exactly at each time step increases by a factor of $K$, the number of cross-product switch settings, where $K = \prod_{m=1}^{M} K^{(m)}$.

In this experiment we use the Gaussian Sum approximation [1]. At each time step we maintain an approximation of $p(\mathbf{x}_t|s_t, \mathbf{y}_{1:t})$ as a mixture of $K$ Gaussians. Calculating the Kalman updates and likelihoods for every possible setting of $s_{t+1}$ will result in the posterior $p(\mathbf{x}_{t+1}|s_{t+1}, \mathbf{y}_{1:t+1})$ having $K^2$ mixture components, which can be collapsed back into $K$ components by matching means and variances of the distribution, as described in [6].

For comparison we also use Rao-Blackwellised particle filtering (RBPF) [7] for approximate inference. In this technique a number of particles are propagated through each time step, each with a switch state $s_t$ and an estimate of the mean and variance of $\mathbf{x}_t$. A value for the switch state $s_{t+1}$ is obtained for each particle by sampling from the transition probabilities, after which Kalman updates are performed and a likelihood value can be calculated. Based on this likelihood, particles can be either discarded or multiplied. Because Kalman updates are not calculated for every possible setting of $s_{t+1}$, this method can give a significant increase in speed when there are many factors, with some tradeoff in accuracy.

Both inference methods can be speeded up by considering the dropout factors. Because a probe dropout always results in an observation of zero on the corresponding measurement channels, the value of $y_t$ can be examined at each step. If it is not equal to zero then we know that the likelihood of a dropout factor being active will be very low, so there is no need to calculate it explicitly. Similarly, if any of the observations are zero then we only perform Kalman updates and calculate likelihoods for those switch states with the appropriate dropout setting.

## 2.3   Relation to previous work

The SKF and various approximations for inference have been described by many authors, see e.g. [10, 3]. In [5], the authors used a 2-factor FSKF in a speech recognition application; the two factors corresponded to (i) phones and (ii) the phone-to-spectrum transformation. There has also been much prior work on condition monitoring in intensive care; here we give a brief review of some of these studies and the relationship to our own work.

The specific problem of artifact detection in physiological time series data has been approached in a number of ways. For example Tsien [9] used machine learning techniques, notably decision trees and logistic regression, to classify each observation $\mathbf{y}_t$ as genuine or artifactual. Hoare and Beatty [4] describe the use of time series analysis techniques

(ARIMA models, moving average and Kalman filters) to predict the next point in a patient's monitoring trace. If the difference between the observed value and the predicted value was outside a predetermined range, the data point was classified as artifactual. Our application of a model with factorial state extends this work by explaining the specific cause of an artifact, rather than just the fact that a certain data point is artifactual or not. We are not aware of other work in condition monitoring using a FSKF.

## 3 Parameter estimation

We use hand-annotated training data from a number of babies to estimate the parameters of the model.

**Factor dynamics**: Using equation 3 we can calculate the state transition probabilities from the transition probabilities for individual state variables, $P(f_t^{(m)} = a | f_{t-1}^{(m)} = b)$. The estimates for these are given by $P(f_t^{(m)} = a | f_{t-1}^{(m)} = b) = (n_{ba} + c) / \left( \sum_{c=1}^{K^{(m)}} (n_{bc} + c) \right)$, where $n_{ba}$ is the number of transitions from state $b$ to state $a$ in the training data. The smoothing constant $c$ (in our experiments we set $c = 1$) is added to stop any of the transition probabilities being zero or very small. While a zero probability could be useful for a sequence of states that we know are impossible, in general we want to avoid it. This solution can be justified theoretically as a maximum a posteriori estimate where the prior is given by a Dirichlet distribution. The factor dynamics can be used to create left-to-right models, e.g. for passing through the sequence $O_2$ high, $CO_2$ low; $CO_2 \rightarrow 0$; equilibration in the TCP recalibration case.

**System dynamics**: When no factor is active (i.e. non-normal), the baby is said to be in a stable condition and has some capacity for self-regulation. In this condition we consider each observation channel separately, and use standard methods to fit AR or ARIMA models to each channel. Most channels vary around reference ranges when the baby is stable and are well fitted by AR(2) models. Heart rate and blood pressure observation channels are more volatile and stationarity is improved after differencing. Heart rate dynamics, for example, are well fitted with an ARIMA(2,1,0) process. Representing trained AR or ARIMA processes in state space form is then straightforward.

The observational data tends to have some high frequency noise on it (see e.g. Fig. 1(b), lower panel) due to probe error and quantization effects. Thus we smooth sections of stable data with a 21-point moving average in order to obtain training data for the system dynamics.

The Yule-Walker equations are then used to set parameters for this moving-averaged data. The fit can be verified for each observation channel by comparing the spectrum of new data with the theoretical spectrum of the AR process (or the spectrum of the differenced data for ARIMA processes), see e.g. [2]. The measurement noise matrix $R$ is estimated by calculating the variance of the differences between the original and averaged training data for each measurement channel.

Above we have modelled the dynamics for a baby in the stable condition; we now describe some of the system models used when the factors are active (i.e. non-normal). The drop and rise in temperature measurements caused by a temperature probe disconnection closely resemble exponential decay and can be therefore be fitted with an AR(1) process. This also applies to the equilibration stage of a TCP recalibration.

The dynamics corresponding to the bradycardia factor are set by finding the mean slope of the fall and rise in heart rate, which is used for the drift term $\mathbf{d}$, then fitting an AR(1) process to the residuals. The arterial blood sample dynamics are modelled with linear drift; note that the variable in $\mathbf{x}_a$ corresponding to the value of the arterial blood sample is tied

|        | Blood sample |       | TCP recal. |       | Bradycardia |       | TC disconnect |       | Incu. open |       |
|--------|:------------:|:-----:|:----------:|:-----:|:-----------:|:-----:|:-------------:|:-----:|:----------:|:-----:|
|        | AUC          | EER   | AUC        | EER   | AUC         | EER   | AUC           | EER   | AUC        | EER   |
| FHMM   | 0.97         | 0.02  | 0.78       | 0.25  | 0.67        | 0.42  | 0.75          | 0.35  | 0.97       | 0.07  |
| GS     | 0.99         | 0.01  | 0.91       | 0.12  | 0.72        | 0.39  | 0.88          | 0.19  | 0.97       | 0.06  |
| RBPF   | 0.62         | 0.46  | 0.90       | 0.14  | 0.76        | 0.37  | 0.85          | 0.32  | 0.95       | 0.08  |

Table 2: Inference results on evaluation data. FHMM denotes the Factorial Hidden Markov Model, GS denotes the Gaussian Sum approximation, and RBPF denotes Rao-Blackwellised particle filtering with 560 particles. AUC denotes area under ROC curve and EER denotes the equal error rate.

to the diastolic blood pressure value while the factor is inactive. We also use linear drift to model the drop in incubator humidity measurements corresponding to a clinician opening the incubator portals.

We assume that the measurement noise from each probe is the same for physiological and artifactual readings, for example if the core temperature probe is attached to the baby's skin or is reading ambient incubator temperature.

**Combining factors**: The parameters $\{A, H, Q, R, \mathbf{d}\}$ have to be supplied for every combination of factors. It might be thought that training data would be needed for each of these possible combinations, but in practice parameters can be trained for factors individually and then combined, as we know that some of the phenomena we want to model only affect a subset of the channels, or override other phenomena [8]. This process of setting parameters for each combination of factor settings can be automated. The factors are arranged in a partially ordered set, where later factors overwrite the dynamics $A, Q, \mathbf{d}$ or observations $H, R$ on at least one channel of their predecessor. For example, the 'bradycardia' factor overwrites the heart rate dynamics of the normal state, while the 'ECG dropout' factor overwrites the heart rate observations; if both these things are happening simultaneously then we expect the same observations as if there was only an ECG dropout, but the dynamics of the true state $\mathbf{x}_p$ are propagated as though there was only a bradycardia. Having found this ordering it is straightforward to merge the trained parameters for every combination of factors.

# 4   Results

Monitoring data was obtained from eight infants of 28 weeks gestation during their first week of life, from the NICU at Edinburgh Royal Infirmary. The data for each infant was collected every second for 24 hours, on nine channels: heart rate, systolic and diastolic blood pressures, TcPO$_2$, TcPCO$_2$, O$_2$ saturation, core temperature and incubator temperature and humidity. These infants were the first 8 in the NICU database who satisfied the age criteria and were monitored on all 8 channels for some 24 hour period within their first week. Four infants were used for training the model and four for evaluation. The test data was annotated with the times of occurrences of the factors in Table 1 by a clinical expert and one of the authors.

Some examples of inference under the model are shown in Figures 1(b) and 2. In Figure 1(b) two factors, arterial blood sample and ECG dropout are simultaneously active, and the inference works nicely in this case, with growing uncertainty about the true value of the heart-rate and blood pressure channels when artifactual readings are observed. The upper panel in figure 2(a) shows two examples of bradycardia being detected. In the lower panel, the model correctly infers the times that a clinician enters the incubator and replaces a disconnected core temperature probe. Figure 2(b) illustrates the simultaneous detection of a TCP artifact (the TCP recal state plotted is obtained by summing the probabilities of

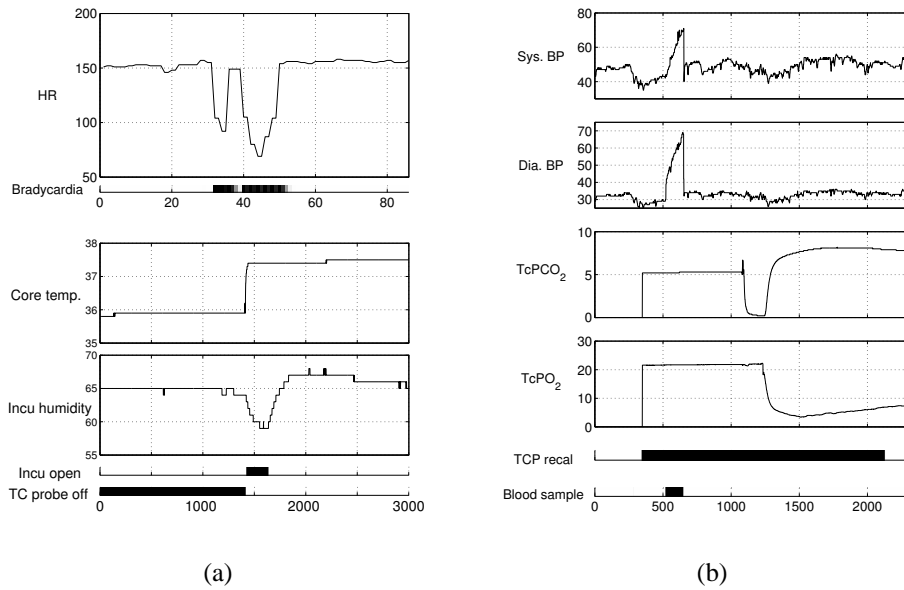

<div align="center">(a)&emsp;&emsp;&emsp;&emsp;&emsp;&emsp;&emsp;&emsp;&emsp;&emsp;&emsp;&emsp;&emsp;&emsp;&emsp;&emsp;(b)</div>

Figure 2: Inferred durations of physiological and artifactual states: (a) shows two episodes of bradycardia *(top)*, and a clinician entering the incubator and replacing the core temperature probe *(bottom)*. Plot (b) shows the inference of two simultaneous artifact processes, arterial blood sampling and TCP recalibration. Times are in seconds.

the three non-normal TCP states) and a blood sample spike.

In Table 2 we show the performance of the model on the test data. The inferred probabilities for each factor were compared with the gold standard which has a binary value for each factor setting at each time point. Inference was done using the Gaussian sum approximation and RBPF, where the number of particles was set so that the two inference methods had the same execution time. As a baseline we also used a Factorial Hidden Markov Model (FHMM) to infer when each factor was active. This model has the same factor structure as the FSKF, without any hidden continuous state. The FHMM parameters were set using the same training data as the FSKF.

It can be seen that the FSKF generalised well to the data from the test set. Inferences using the Gaussian Sum approximation had consistently higher area under the ROC curve and lower equal error rates than the FHMM. In particular, the inferred times of blood samples and incubator opening were reliably detected. The lower performance of the FHMM, which has no knowledge of the dynamics, illustrates the difficulty caused by baseline physiological levels changing over time and between babies.

Inference results using Rao-Blackwellised particle filtering were less consistent. For blood sampling and opening of the incubator the performance was worse than the baseline model, though in detecting bradycardia the performance was marginally higher than for inferences made using either the FHMM or the Gaussian Sum approximation.

Execution times for inference on 24 hours of monitoring data with the set of factors listed in Table 1 on a 3.2GHz processor were approximately 7 hours 10 minutes for the FSKF inference, and 100 seconds for the FHMM.

# 5 Discussion

In this paper we have shown that the FSKF model can be applied successfully to complex monitoring data from a neonatal intensive care unit.

There are a number of directions in which this work can be extended. Firstly, for simplicity we have used univariate autoregressive models for each component of the observations; it would be interesting to fit a multivariate model to this data instead, as we expect that there will be correlations between the channels. Also, there are additional factors that can be incorporated into the model, for example to model a pneumothorax event, where air becomes trapped inside the chest between the chest wall and the lung, causing the lung to collapse. Fortunately this event is relatively rare so it was not seen in the data we have analyzed in this experiment.

### Acknowledgements

We thank Birgit Wefers for providing expert annotation of the evaluation data set, and the anonymous referees for their comments which helped improve the paper. This work was funded in part by a grant from the premature baby charity BLISS. The work was also supported in part by the IST Programme of the European Community, under the PAS-CAL Network of Excellence, IST-2002-506778. This publication only reflects the authors' views.

# References

[1] D. L. Alspach and H. W. Sorenson. Nonlinear Bayesian Estimation Using Gaussian Sum Approximations. *IEEE Transactions on Automatic Control*, 17(4):439–448, 1972.

[2] C. Chatfield. *The Analysis of Time Series: An Introduction*. Chapman and Hall, London, 4th edition, 1989.

[3] Z. Ghahramani and G. E. Hinton. Variational Learning for Switching State-Space Models. *Neural Computation*, 12(4):963–996, 1998.

[4] S.W. Hoare and P.C.W. Beatty. Automatic artifact identification in anaesthesia patient record keeping: a comparison of techniques. *Medical Engineering and Physics*, 22:547–553, 2000.

[5] J. Ma and L. Deng. A mixed level switching dynamic system for continuous speech recognition. *Computer Speech and Language*, 18:49–65, 2004.

[6] K. Murphy. Switching Kalman filters. Technical report, U.C. Berkeley, 1998.

[7] K. Murphy and S. Russell. Rao-Blackwellised particle filtering for dynamic Bayesian networks. In A. Doucet, N. de Freitas, and N. Gordon, editors, *Sequential Monte Carlo in Practice*. Springer-Verlag, 2001.

[8] A. Spengler. Neonatal baby monitoring. Master's thesis, School of Informatics, University of Edinburgh, 2003.

[9] C. Tsien. *TrendFinder: Automated Detection of Alarmable Trends*. PhD thesis, MIT, 2000.

[10] M. West and P. J. Harrison. *Bayesian Forecasting and Dynamic Models*. Springer-Verlag, 1997. Second edition.
